# Adaptive Quantization and Density Estimation in Silicon

**David Hsu      Seth Bridges      Miguel Figueroa      Chris Diorio**
Department of Computer Science and Engineering
University of Washington
114 Sieg Hall, Box 352350
Seattle, WA 98195-2350 USA
*{hsud, seth, miguel,  diorio}@cs.washington.edu*

## Abstract

We present the bump mixture model, a statistical model for analog
data where the probabilistic semantics, inference, and learning
rules derive from low-level transistor behavior. The bump mixture
model relies on translinear circuits to perform probabilistic infer-
ence, and floating-gate devices to perform adaptation. This system
is low power, asynchronous, and fully parallel, and supports vari-
ous on-chip learning algorithms. In addition, the mixture model can
perform several tasks such as probability estimation, vector quanti-
zation, classification, and clustering. We tested a fabricated system
on clustering, quantization, and classification of handwritten digits
and show performance comparable to the E-M algorithm on mix-
tures of Gaussians.

## 1   Introduction

Many system-on-a-chip applications, such as data compression and signal process-
ing, use online adaptation to improve or tune performance. These applications can
benefit from the low-power compact design that analog VLSI learning systems can
offer. Analog VLSI learning systems can benefit immensely from flexible learning
algorithms that take advantage of silicon device physics for compact layout, and that
are capable of a variety of learning tasks. One learning paradigm that encompasses a
wide variety of learning tasks is density estimation, learning the probability
distribution over the input data. A silicon density estimator can provide a basic
template for VLSI systems for feature extraction, classification, adaptive vector
quantization, and more.

In this paper, we describe the bump mixture model, a statistical model that describes
the probability distribution function of analog variables using low-level transistor
equations. We intend the bump mixture model to be the silicon version of mixture of
Gaussians [1], one of the most widely used statistical methods for modeling the
probability distribution of a collection of data. Mixtures of Gaussians appear in
many contexts from radial basis functions [1] to hidden Markov models [2]. In the
bump mixture model, probability computations derive from translinear circuits [3]
and learning derives from floating-gate device equations [4]. The bump mixture

model can perform different functions such as quantization, probability estimation, and classification. In addition this VLSI mixture model can implement multiple learning algorithms using different peripheral circuitry. Because the equations for system operation and learning derive from natural transistor behavior, we can build large bump mixture model with millions of parameters on a single chip. We have fabricated a bump mixture model, and tested it on clustering, classification, and vector quantization of handwritten digits. The results show that the fabricated system performs comparably to mixtures of Gaussians trained with the E-M algorithm [1].

Our work builds upon several trends of research in the VLSI community. The results in this paper are complement recent work on probability propagation in analog VLSI [5-7]. These previous systems, intended for decoding applications in communication systems, model special forms of probability distributions over discrete variables, and do not incorporate learning. In contrast, the bump mixture model performs inference and learning on probability distributions over continuous variables. The bump mixture model significantly extends previous results on floating-gate circuits [4]. Our system is a fully realized floating-gate learning algorithm that can be used for vector quantization, probability estimation, clustering, and classification. Finally, the mixture model's architecture is similar to many previous VLSI vector quantizers [8, 9]. We can view the bump mixture model as a VLSI vector quantizer with well-defined probabilistic semantics. Computations such as probability estimation and maximum-likelihood classification have a natural statistical interpretation under the mixture model. In addition, because we rely on floating-gate devices, the mixture model does not require a refresh mechanism unlike previous learning VLSI quantizers.

## 2   The adaptive bump circuit

The adaptive bump circuit [4], depicted in Fig. 1(a-b), forms the basis of the bump mixture model. This circuit is slightly different from previous versions reported in the literature. Nevertheless, the high level functionality remains the same; the adaptive bump circuit computes the similarity between a stored variable and an input, and adapts to increase the similarity between the stored variable and input.

Fig. 1(a) shows the computation portion of the circuit. The bump circuit takes as input, a differential voltage signal $(+V_{in}, -V_{in})$ around a DC bias, and computes the similarity between $V_{in}$ and a stored value, $\mu$. We represent the stored memory $\mu$ as a voltage:

$$\mu = \frac{V_{w-} - V_{w+}}{2} \tag{1}$$

where $V_{w+}$ and $V_{w-}$ are the gate-offset voltages stored on capacitors $C_1$ and $C_2$. Because $C_1$ and $C_2$ isolate the gates of transistors $M_1$ and $M_2$ respectively, these transistors are floating-gate devices. Consequently, the stored voltages $V_{w+}$ and $V_{w-}$ are nonvolatile. We can express the floating-gate voltages $V_{fg1}$ and $V_{fg2}$ as $V_{fg1} = V_{in} + V_{w+}$ and $V_{fg2} = V_{w-} - V_{in}$, and the output of the bump circuit as [10]:

$$I_{out} = \frac{I_b}{\cosh^2\left(\left(4\kappa/SU_t\right)\left(V_{fg1} - V_{fg2}\right)\right)} = \frac{I_b}{\cosh^2\left(\left(8\kappa/SU_t\right)\left(V_{in} - \mu\right)\right)} \tag{2}$$

where $I_b$ is the bias current, $\kappa$ is the gate-coupling coefficient, $U_t$ is the thermal voltage, and $S$ depends on the transistor sizes. Fig. 1(b) shows $I_{out}$ for three different stored values of $\mu$. As the data show, different $\mu$'s shift the location of the peak response of the circuit.

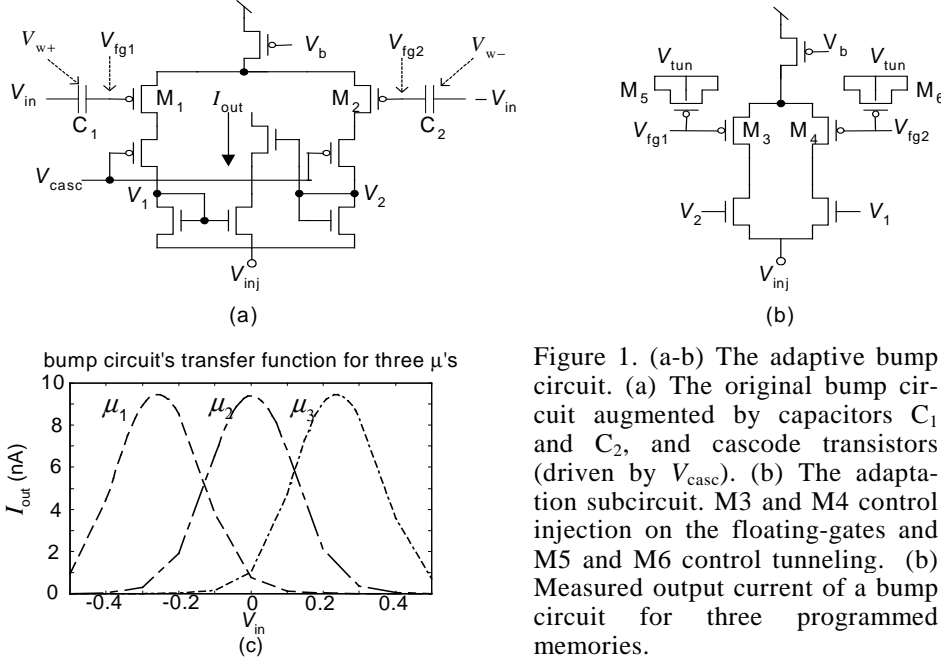

(a)                  (b)

bump circuit's transfer function for three μ's

Figure 1. (a-b) The adaptive bump circuit. (a) The original bump circuit augmented by capacitors $C_1$ and $C_2$, and cascode transistors (driven by $V_{casc}$). (b) The adaptation subcircuit. M3 and M4 control injection on the floating-gates and M5 and M6 control tunneling. (b) Measured output current of a bump circuit for three programmed memories.

(c)

Fig.1(b) shows the circuit that implements learning in the adaptive bump circuit. We implement learning through Fowler-Nordheim tunneling [11] on tunneling junctions $M_5$-$M_6$ and hot electron injection [12] on the floating-gate transistors $M_3$-$M_4$. Transistor $M_3$ and $M_5$ control injection and tunneling on $M_1$'s floating-gate. Transistors $M_4$ and $M_6$ control injection and tunneling on $M_2$'s floating-gate. We activate tunneling and injection by a high $V_{tun}$ and low $V_{inj}$ respectively. In the adaptive bump circuit, both processes increase the similarity between $V_{in}$ and $\mu$. In addition, the magnitude of the update does not depend on the sign of $(V_{in} - \mu)$ because the differential input provides common-mode rejection to the input differential pair.

The similarity function, as seen in Fig.1(b), has a Gaussian-like shape. Consequently, we can equate the output current of the bump circuit with the probability of the input under a distribution parameterized by mean $\mu$:

$$P\left(V_{in} \mid \mu\right) = I_{out} \tag{3}$$

In addition, increasing the similarity between $V_{in}$ and $\mu$ is equivalent to increasing $P(V_{in} \mid \mu)$. Consequently, the adaptive bump circuit adapts to maximize the likelihood of the present input under the circuit's probability distribution.

## 3   The bump mixture model

We now describe the computations and learning rule implemented by the bump mixture model. A mixture model is a general class of statistical models that approximates the probability of an analog input as the weighted sum of probability of the input under several simple distributions. The bump mixture model comprises a set of Gaussian-like probability density functions, each parameterized by a mean vector, $\boldsymbol{\mu_i}$. Denoting the $j^{th}$ dimension of the mean of the $i^{th}$ density as $\mu_{ij}$, we express the probability of an input vector $\boldsymbol{x}$ as:

$$P(\pmb{x}) = (1/N)\sum_i P(\pmb{x}\,|\,i) = (1/N)\sum_i \left(\prod_j P\left(x_j\,|\,\mu_{ij}\right)\right) \tag{4}$$

where $N$ is the number of densities in the model and $i$ denotes the i[th] density. $P(\pmb{x}|i)$ is the product of one-dimensional densities $P(x_j|\mu_{ij})$ that depend on the j[th] dimension of the i[th] mean, $\mu_{ij}$. We derive each one-dimensional probability distribution from the output current of a single bump circuit. The bump mixture model makes two assumptions: (1) the component densities are equally likely, and (2) within each component density, the input dimensions are independent and have equal variance. Despite these restrictions, this mixture model can, in principle, approximate any probability density function [1].

The bump mixture model adapts all $\pmb{\mu}_i$ to maximize the likelihood of the training data. Learning in the bump mixture model is based on the E-M algorithm, the standard algorithm for training Gaussian mixture models. The E-M algorithm comprises two steps. The E-step computes the conditional probability of each density given the input, $P(i|\pmb{x})$. The M-step updates the parameters of each distribution to increase the likelihood of the data, using $P(i|\pmb{x})$ to scale the magnitude of each parameter update. In the online setting, the learning rule is:

$$\Delta\mu_{ij} = \eta P(i\,|\,\pmb{x})\frac{\partial \log P\left(x_j\,|\,\mu_{ij}\right)}{\partial \mu_{ij}} = \eta \frac{P(\pmb{x}\,|\,i)}{\sum_k P(\pmb{x}\,|\,k)}\frac{\partial \log P\left(x_j\,|\,\mu_{ij}\right)}{\partial \mu_{ij}} \tag{5}$$

where $\eta$ is a learning rate and $k$ denotes component densities. Because the adaptive bump circuit already adapts to increase the likelihood of the present input, we approximate E-M by modulating injection and tunneling in the adaptive bump circuit by the conditional probability:

$$\Delta\mu_{ij} = \eta P\left(i\,|\,\pmb{x}\right)f\left(x_j - \mu_{ij}\right) \tag{6}$$

where $f()$ is the parameter update implemented by the bump circuit. We can modulate the learning update in (6) with other competitive factors instead of the conditional probability to implement a variety of learning rules such as online K-means.

## 4   Silicon implementation

We now describe a VLSI system that implements the silicon mixture model. The high level organization of the system detailed in Fig.2, is similar to VLSI vector quantization systems. The heart of the mixture model is a matrix of adaptive bump circuits where the i[th] row of bump circuits corresponds to the i[th] component density. In addition, the periphery of the matrix comprises a set of inhibitory circuits for performing probability estimation, inference, quantization, and generating feedback for learning.

We send each dimension of an input $\pmb{x}$ down a single column. Unity-gain inverting amplifiers (not pictured) at the boundary of the matrix convert each single ended voltage input into a differential signal. Each bump circuit computes a current that represents $(P(x_j|\mu_{ij}))^\sigma$, where $\sigma$ is the common variance of the one-dimensional densities. The mixture model computes $P(\pmb{x}|i)$ along the i[th] row and inhibitory circuits perform inference, estimation, or quantization. We utilize translinear devices [3] to perform all of these computations. Translinear devices, such as the subthreshold MOSFET and bipolar transistor, exhibit an exponential relationship between the gate-voltage and source current. This property allows us to establish a power-law relationship between currents and probabilities (i.e. a linear relationship between gate voltages and log-probabilities).

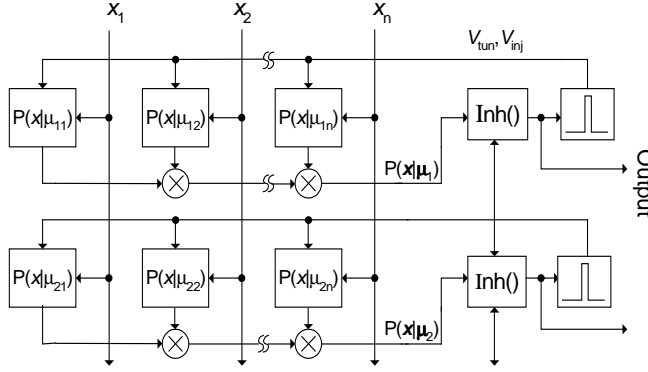

Figure 2. Bump mixture model architecture. The system comprises a matrix of adaptive bump circuits where each row computes the probability $P(\boldsymbol{x}|\boldsymbol{\mu}_i)$. Inhibitory circuits transform the output of each row into system outputs. Spike generators also transform inhibitory circuit outputs into rate-coded feedback for learning.

We compute the multiplication of the probabilities in each row of Fig.2 as addition in the log domain using the circuit in Fig.3(a). This circuit first converts each bump circuit's current into a voltage using a diode (e.g. $M_1$). $M_2$'s capacitive divider computes $V_{avg}$ as the average of the scalar log probabilities, $\log P(x_j|\mu_{ij})$:

$$V_{avg} = (\sigma / N)\sum_{j}\log P\left(x_j \mid \mu_{ij}\right) \tag{7}$$

where $\sigma$ is the variance, $N$ is the number of input dimensions, and voltages are in units of $\kappa/U_t$ ($U_t$ is the thermal voltage and $\kappa$ is the transistor-gate coupling coefficient). Transistors $M_2$- $M_5$ mirror $V_{avg}$ to the gate of $M_5$. We define the drain voltage of $M_5$ as $\log P(\boldsymbol{x}|i)$ (up to an additive constant) and compute:

$$\log\left(P\left(\boldsymbol{x} \mid i\right)\right) = \frac{(C_1+C_2)}{C_1}V_{avg} = \frac{(C_1+C_2)\sigma}{C_1 N}\sum_{j}\log\left(P\left(x_j \mid \mu_{ij}\right)\right)+k \tag{8}$$

where $k$ is a constant dependent on $V_g$ (the control gate voltage on $M_5$), and $C_1$ and $C_2$ are capacitances. From eq.8 we can derive the variance as:

$$\sigma = NC_1 /\left(C_1 + C_2\right) \tag{9}$$

The system computes different output functions and feedback signals for learning by operating on the log probabilities of eq.8. Fig.3(b) demonstrates a circuit that computes $P(i|\boldsymbol{x})$ for each distribution. The circuit is a k-input differential pair where the bias transistor $M_0$ normalizes currents representing the probabilities $P(\boldsymbol{x}|i)$ at the $i^{th}$ leg. Fig.3(c) demonstrates a circuit that computes $P(\boldsymbol{x})$. The $i^{th}$ transistor exponentiates $\log P(\boldsymbol{x}|i)$, and a single wire sums the currents. We can also apply other inhibitory circuits to the log probabilities such as winner-take-all circuits (WTA) [13] and resistive networks [14]. In our fabricated chip, we implemented probability estimation, conditional probability computation, and WTA. The WTA outputs the index of the most likely component distribution for the present input, and can be used to implement vector quantization and to produce feedback for an online K-means learning rule.

At each synapse, the system combines a feedback signal, such as the conditional probability $P(i|\boldsymbol{x})$, computed at the matrix periphery, with the adaptive bump circuit to implement learning. We trigger adaptation at each bump circuit by a rate-coded spike signal generated from the inhibitory circuit's current outputs. We generate this spike train with a current-to-spike converter based on Lazzaro's low-powered spiking neuron [15]. This rate-coded signal toggles $V_{tun}$ and $V_{inj}$ at each bump circuit. Consequently, adaptation is proportional to the frequency of the spike train, which is in turn a linear function of the inhibitory feedback signal. The alternative to the rate code would be to transform the inhibitory circuit's output directly into analog

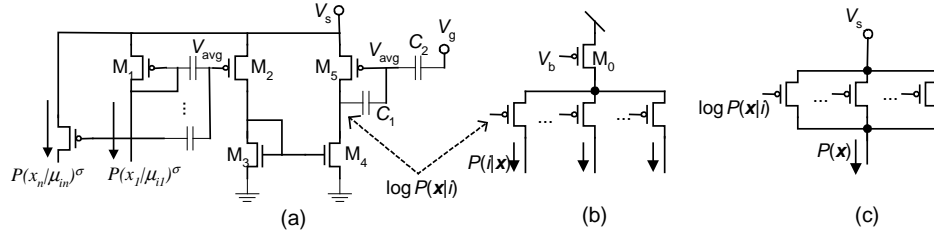

Figure 3. (a) Circuit for computing $\log P(\boldsymbol{x}|i)$. (b) Circuit for computing $P(i|\boldsymbol{x})$. The current through the $i^{\text{th}}$ leg represents $P(i|\boldsymbol{x})$. (c) Circuit for computing $P(\boldsymbol{x})$.

$V_{\text{tun}}$ and $V_{\text{inj}}$ signals. Because injection and tunneling are highly nonlinear functions of $V_{\text{inj}}$ and $V_{\text{tun}}$ respectively, implementing updates that are linear in the inhibitory feedback signal is quite difficult using this approach.

## 5  Experimental Results and Conclusions

We fabricated an 8 x 8 mixture model (8 probability distribution functions with 8 dimensions each) in a TSMC 0.35$\mu$m CMOS process available through MOSIS, and tested the chip on synthetic data and a handwritten digits dataset. In our tests, we found that due to a design error, one of the input dimensions coupled to the other inputs. Consequently, we held that input fixed throughout the tests, effectively reducing the input to 7 dimensions. In addition, we found that the learning rule in eq. 6 produced poor performance because the variance of the bump distributions was too large. Consequently, in our learning experiments, we used the hard winner-take-all circuit to control adaptation, resulting in a K-means learning rule. We trained the chip to perform different tasks on handwritten digits from the MNIST dataset [16]. To prepare the data, we first perform PCA to reduce the 784-pixel images to seven-dimensional vectors, and then sent the data on-chip.

We first tested the circuit on clustering handwritten digits. We trained the chip on 1000 examples of each of the digits 1-8. Fig. 4(a) shows reconstructions of the eight means before and after training. We compute each reconstruction by multiplying the means by the seven principal eigenvectors of the dataset. The data shows that the means diverge to associate with different digits. The chip learns to associate most digits with a single probability distribution. The lone exception is digit 5 which doesn't clearly associate with one distribution. We speculate that the reason is that 3's, 5's, and 8's are very similar in our training data's seven-dimensional representation. Gaussian mixture models trained with the E-M algorithm also demonstrate similar results, recovering only seven out of the eight digits.

We next evaluated the same learned means on vector quantization of a set of test digits (4400 examples of each digit). We compare the chip's learned means with means learned by the batch E-M algorithm on mixtures of Gaussians (with $\sigma$=0.01), a mismatch E-M algorithm that models chip nonidealities, and a non-adaptive baseline quantizer. The purpose of the mismatch E-M algorithm was to assess the effect of nonuniform injection and tunneling strengths in floating-gate transistors. Because tunneling and injection magnitudes can vary by a large amount on different floating-gate transistors, the adaptive bump circuits can learn a mean that is somewhat off-center. We measured the offset of each bump circuit when adapting to a constant input and constructed the mismatch E-M algorithm by altering the learned means during the M-step by the measured offset. We constructed the baseline quantizer by selecting, at random, an example of each digit for the quantizer codebook. For each quantizer, we computed the reconstruction error on the digit's seven-dimensional

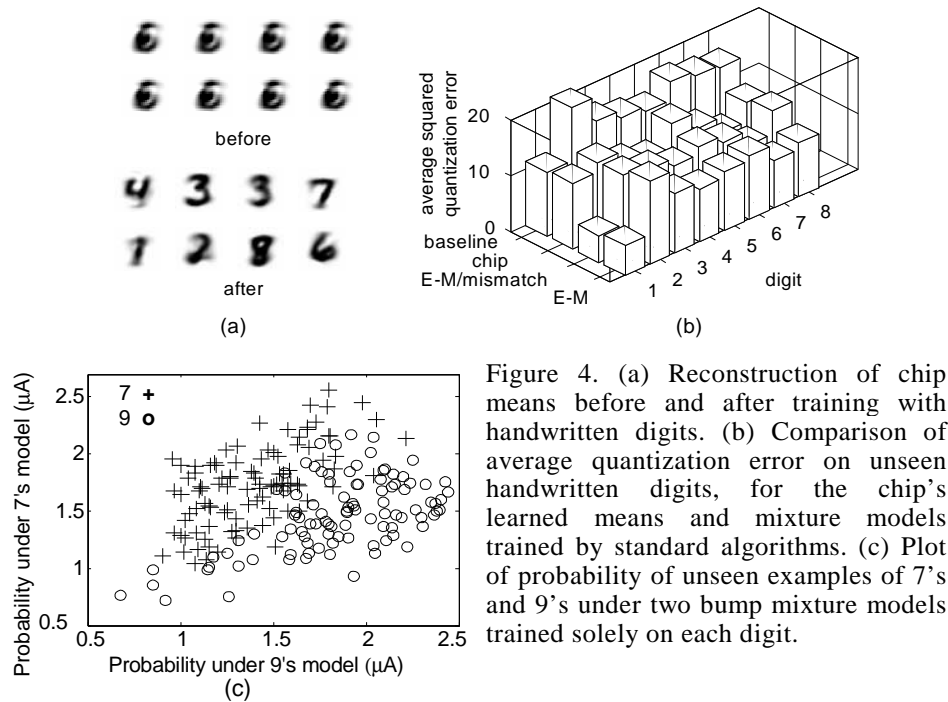

(a)

(b)

Figure 4. (a) Reconstruction of chip means before and after training with handwritten digits. (b) Comparison of average quantization error on unseen handwritten digits, for the chip's learned means and mixture models trained by standard algorithms. (c) Plot of probability of unseen examples of 7's and 9's under two bump mixture models trained solely on each digit.

(c)

representation when we represent each test digit by the closest mean. The results in Fig.4(b) show that for most of the digits the chip's learned means perform as well as the E-M algorithm, and better than the baseline quantizer in all cases. The one digit where the chip's performance is far from the E-M algorithm is the digit "1". Upon examination of the E-M algorithm's results, we found that it associated two means with the digit "1", where the chip allocated two means for the digit "3". Over all the digits, the E-M algorithm exhibited a quantization error of 9.98, mismatch E-M gives a quantization error of 10.9, the chip's error was 11.6, and the baseline quantizer's error was 15.97. The data show that mismatch is a significant factor in the difference between the bump mixture model's performance and the E-M algorithm's performance in quantization tasks.

Finally, we use the mixture model to classify handwritten digits. If we train a separate mixture model for each class of data, we can classify an input by comparing the probabilities of the input under each model. In our experiment, we train two separate mixture models: one on examples of the digit 7, and the other on examples of the digit 9. We then apply both mixtures to a set of unseen examples of digits 7 and 9, and record the probability score of each unseen example under each mixture model. We plot the resulting data in Fig.4(c). Each axis represents the probability under a different class. The data show that the model probabilities provide a good metric for classification. Assigning each test example to the class model that outputs the highest probability results in an accuracy of 87% on 2000 unseen digits. Additional software experiments show that mixtures of Gaussians ($\sigma$=0.01) trained by the batch E-M algorithm provide an accuracy of 92.39% on this task.

Our test results show that the bump mixture model's performance on several learning tasks is comparable to standard mixtures of Gaussians trained by E-M. These experiments give further evidence that floating-gate circuits can be used to build effective learning systems even though their learning rules derive from silicon physics instead of statistical methods. The bump mixture model also represents a basic building block that we can use to build more complex silicon probability models

over analog variables. This work can be extended in several ways. We can build distributions that have parameterized covariances in addition to means. In addition, we can build more complex, adaptive probability distributions in silicon by combining the bump mixture model with silicon probability models over discrete variables [5-7] and spike-based floating-gate learning circuits [4].

## Acknowledgments

This work was supported by NSF under grants BES 9720353 and ECS 9733425, and Packard Foundation and Sloan Fellowships.

## References

[1]     C. M. Bishop, *Neural Networks for Pattern Recognition*. Oxford, UK: Clarendon Press, 1995.

[2]     L. R. Rabiner, "A tutorial on hidden Markov models and selected applications in speech recognition," *Proceedings of the IEEE*, vol. 77, pp. 257-286, 1989.

[3]     B. A. Minch, "Analysis, Synthesis, and Implementation of Networks of Multiple-Input Translinear Elements," California Institute of Technology, 1997.

[4]     C.Diorio, D.Hsu, and M.Figueroa, "Adaptive CMOS: from biological inspiration to systems-on-a-chip," *Proceedings of the IEEE*, vol. 90, pp. 345-357, 2002.

[5]     T. Gabara, J. Hagenauer, M. Moerz, and R. Yan, "An analog 0.25 μm BiCMOS tail-biting MAP decoder," IEEE International Solid State Circuits Conference (ISSCC), 2000.

[6]     J. Dai, S. Little, C. Winstead, and J. K. Woo, "Analog MAP decoder for (8,4) Hamming code in subthreshold CMOS," Advanced Research in VLSI (ARVLSI), 2001.

[7]     M. Helfenstein, H.-A. Loeliger, F. Lustenberger, and F. Tarkoy, "Probability propagation and decoding in analog VLSI," *IEEE Transactions on Information Theory*, vol. 47, pp. 837-843, 2001.

[8]     W. C. Fang, B. J. Sheu, O. Chen, and J. Choi, "A VLSI neural processor for image data compression using self-organization neural networks," *IEEE Transactions on Neural Networks*, vol. 3, pp. 506-518, 1992.

[9]     J. Lubkin and G. Cauwenberghs, "A learning parallel analog-to-digital vector quantizer," *Journal of Circuits, Systems, and Computers*, vol. 8, pp. 604-614, 1998.

[10]    T. Delbruck, "Bump circuits for computing similarity and dissimilarity of analog voltages," California Institute of Technology, CNS Memo 26, 1993.

[11]    M. Lenzlinger, and E. H. Snow, "Fowler-Nordheim tunneling into thermally grown $SiO_2$," *Journal of Applied Physics*, vol. 40, pp. 278-283, 1969.

[12]    E. Takeda, C. Yang, and A. Miura-Hamada, *Hot Carrier Effects in MOS Devices*. San Diego, CA: Academic Press, 1995.

[13]    J. Lazzaro, S. Ryckebusch, M. Mahowald, and C. A. Mead, "Winner-take-all networks of O(n) complexity," in *Advances in Neural Information Processing*, vol. 1, D. Tourestzky, Ed.: MIT Press, 1989, pp. 703-711.

[14]    K. Boahen and A. Andreou, "A contrast sensitive silicon retina with reciprocal synapses," in *Advances in Neural Information Processing Systems 4*, S. H. J. Moody, and R. Lippmann, Ed.: MIT Press, 1992, pp. 764-772.

[15]    J. Lazzaro, "Low-power silicon spiking neurons and axons," IEEE International Symposium on Circuits and Systems, 1992.

[16]    Y. Lecun, "The MNIST database of handwritten digits, http://yann_lecun.com/exdb/mnist."
